# A Study of Parallel Perturbative Gradient Descent

**D. Lippe* J. Alspector**
Bellcore
Morristown, NJ 07960

## Abstract

We have continued our study of a parallel perturbative learning method [Alspector et al., 1993] and implications for its implementation in analog VLSI. Our new results indicate that, in most cases, a single parallel perturbation (per pattern presentation) of the function parameters (weights in a neural network) is theoretically the best course. This is not true, however, for certain problems and may not generally be true when faced with issues of implementation such as limited precision. In these cases, multiple parallel perturbations may be best as indicated in our previous results.

## 1  INTRODUCTION

Motivated by difficulties in analog VLSI implementation of back-propagation [Rumelhart et al., 1986] and related algorithms that calculate gradients based on detailed knowledge of the neural network model, there were several similar recent papers proposing to use a parallel [Alspector et al., 1993, Cauwenberghs, 1993, Kirk et al., 1993] or a semi-parallel [Flower and Jabri, 1993] perturbative technique which has the property that it measures (with the physical neural network) rather than calculates the gradient. This technique is closely related to methods of stochastic approximation [Kushner and Clark, 1978] which have been investigated recently by workers in fields other than neural networks. [Spall, 1992] showed that averaging multiple parallel perturbations for each pattern presentation may be asymptotically preferable in the presence of noise. Our own results [Alspector et al., 1993] indicated

that multiple parallel perturbations are also preferable when only limited precision is available in the learning rate which is realistic for a physical implementation. In this work we have investigated whether multiple parallel perturbations for each pattern are non-asymptotically preferable theoretically (without noise). We have also studied this empirically, to the limited degree that simulations allow, by removing the precision constraints of our previous work.

## 2   GRADIENT ESTIMATION BY PARALLEL WEIGHT PERTURBATION

Following our previous work, one can estimate the gradient of the error, $E(\vec{w})$, with respect to any weight, $w_i$, by perturbing $w_i$ by $\delta w_i$ and measuring the change in the output error, $\delta E$, as the entire weight vector, $\vec{w}$, except for component $w_i$ is held constant.

$$\frac{\delta E}{\delta w_i} = \frac{E(\vec{w} + \delta \vec{w_i}) - E(\vec{w})}{\delta w_i}$$

We now consider perturbing all weights simultaneously. However, we wish to have the perturbation vector, $\delta \vec{w}$, chosen uniformly on a hypercube. Note that this requires only a random sign multiplying a fixed perturbation and is natural for VLSI using a parallel noise generator [Alspector et al., 1991].

This leads to the approximation (ignoring higher order terms)

$$\frac{\delta E}{\delta w_i} = \frac{\partial E}{\partial w_i} + \sum_{j \neq i}^{W} \left( \frac{\partial E}{\partial w_j} \right) \left( \frac{\delta w_j}{\delta w_i} \right). \tag{1}$$

The last term has expectation value zero for random and independently distributed $\delta w_i$. The weight change rule

$$\Delta w_i = -\eta \frac{\delta E}{\delta w_i},$$

where $\eta$ is a learning rate, will follow the gradient on the average but with considerable noise.

For each pattern, one can reduce the variance of the noise term in (1) by repeating the random parallel perturbation many times to improve the statistical estimate. If we average over $P$ perturbations, we have

$$\frac{\delta E}{\delta w_i} = \frac{1}{P} \sum_{\rho=1}^{P} \frac{\delta E^{(\rho)}}{\delta w_i^{(\rho)}} = \frac{\partial E}{\partial w_i} + \frac{1}{P} \sum_{\rho=1}^{P} \sum_{j \neq i}^{W} \left( \frac{\partial E}{\partial w_j} \right) \left( \frac{\delta w_j^{(\rho)}}{\delta w_i^{(\rho)}} \right)$$

where $\rho$ indexes the perturbation number.

## 3  THEORETICAL RELATIVE EFFICIENCY

### 3.1  BACKGROUND

Spall [Spall, 1992] shows in an asymptotic sense that multiple perturbations may be faster if only a noisy measurement of $E(\vec{w})$ is available, and that one perturbation is superior otherwise. His results are asymptotic in that they compare the rate of convergence to the local minimum if the algorithms run for infinite time. Thus, his results may only indicate that 1 perturbation is superior close to a local minimum. Furthermore, his result implicitly assumes that $P$ perturbations per weight update takes $P$ times as long as 1 perturbation per weight update. Experience shows that the time required to present patterns to the hardware is often the bottleneck in VLSI implementations of neural networks [Brown et al., 1992]. In a hardware implementation of a perturbative learning algorithm, a few perturbations might be performed with no time penalty while waiting for the next pattern presentation.

The remainder of this section sketches an argument that multiple perturbations may be desirable for some problems in a non-asymptotic sense, even in a noise free environment and under the assumption of a multiplicative time penalty for performing multiple perturbations. On the other hand, the argument also shows that there is little reason to believe in practice that any given problem will be learned more quickly by multiple perturbations. Space limitations prevent us from reproducing the full argument and discussion of its relevance which can be found in [Lippe, 1994].

The argument fixes a point in weight space and considers the expectation value of the change in the error induced by one weight update under both the 1 perturbation case and the multiple perturbation case. [Cauwenberghs, 1994] contains a somewhat related analysis of the relative speed of one parallel perturbation and weight perturbation as described in [Jabri and Flower, 1991]. The analysis is only truly relevant far from a local minimum because close to a local minimum the variance of the change of the error is as important as the mean of the change of the error.

### 3.2  Calculations

If $P$ is the number of perturbations, then our learning rule is

$$\Delta w_i = \frac{-\eta}{P} \sum_{\rho=1}^{P} \frac{\delta E^{(\rho)}}{\delta w_i^{(\rho)}}. \tag{2}$$

If $W$ is the number of weights, then $\Delta E$, calculated to second order in $\eta$, is

$$\Delta E = \sum_{i=1}^{W} \frac{\partial E}{\partial w_i} \Delta w_i + \frac{1}{2} \sum_{i=1}^{W} \sum_{j=1}^{W} \frac{\partial^2 E}{\partial w_i \partial w_j} \Delta w_i \Delta w_j. \tag{3}$$

Expanding $\delta E^{(\rho)}$ to second order in $\sigma$ (where $\delta w_i = \pm\sigma$), we obtain

$$\delta E^{(\rho)} = \sum_{j=1}^{W} \frac{\partial E}{\partial w_j} \delta w_j^{(\rho)} + \frac{1}{2} \sum_{j=1}^{W} \sum_{k=1}^{W} \frac{\partial^2 E}{\partial w_j \partial w_k} \delta w_j^{(\rho)} \delta w_k^{(\rho)}. \tag{4}$$

[Lippe, 1994] shows that combining (2)-(4), retaining only first and second order terms, and taking expectation values gives

$$< \Delta E >= -\eta X + \frac{\eta^2}{P}(Y + PZ) \tag{5}$$

where

$$X = \sum_{i=1}^{W} \left(\frac{\partial E}{\partial w_i}\right)^2,$$

$$Z = \frac{1}{2}\sum_{i,k=1}^{W} \frac{\partial^2 E}{\partial w_i \partial w_k} \frac{\partial E}{\partial w_i} \frac{\partial E}{\partial w_k},$$

$$Y = Z + \frac{1}{2}\sum_{i,k=1}^{W} \frac{\partial^2 E}{\partial w_i^2} \left(\frac{\partial E}{\partial w_k}\right)^2 - \sum_{i=1}^{W} \frac{\partial^2 E}{\partial w_i^2} \left(\frac{\partial E}{\partial w_i}\right)^2,$$

Note that first term in (5) is strictly less than or equal to 0 since $X$ is a sum of squares[1]. The second term, on the other hand, can be either positive or negative. Clearly then a sufficient condition for learning is that the first term dominates the second term. By making $\eta$ small enough, we can guarantee that learning occurs. Strictly speaking, this is not a necessary condition for learning. However, it is important to keep in mind that we are only focusing on one point in weight space. If, at this point in weight space, $< \Delta E >$ is negative but the second term's magnitude is close to the first term's magnitude, it is not unlikely that at some other point in weight space $< \Delta E >$ will be positive. Thus, we will assume that for efficient learning to occur, it is necessary that $\eta$ be small enough to make the first term dominate the second term.

Assume that some problem can be successfully learned with one perturbation, at learning rate $\eta(1)$. Then the first order term in (5) dominates the second order terms. Specifically, at any point in weight space we have, for some large constant $\mu$,

$$\eta(1)X \geq \mu\eta(1)^2|Y + Z|$$

In order to learn with $P$ perturbations, we apparently need

$$\eta(P)X \geq \mu\frac{\eta(P)^2}{P}|Y + PZ| \tag{6}$$

The assumption that the first order term of (5) dominates the second order terms implies that convergence time is proportional to $\frac{P}{\eta(P)}$. Thus, learning is more efficient in the multiple perturbation case if

$$\frac{\mu\eta(P)}{P} > \mu\eta(1) \tag{7}$$

It turns out, as shown in [Lippe, 1994] that the conditions (6) and (7) can be met simultaneously with multiple perturbations if $\frac{-Y}{Z} \geq 2$.

It is shown in [Lippe, 1994], by using the fact that the Hessian of a quadratic function with a minimum is positive semi-definite, that if E is quadratic and has a minimum, then $Y$ and $Z$ have the same sign (and hence $\frac{-Y}{Z} < 2$). Any well behaved function acts quadratically sufficiently close to a stationary point. Thus, we can not get $< \Delta E >$ more than a factor of $P$ larger by using $P$ perturbations near local minima of well behaved functions. Although, as mentioned earlier, we are entirely ignoring the issue of the variance of $\Delta E$, this may be some indication of the asymptotic superiority of 1 perturbation.

### 3.3 Discussion of Results

The result that multiple perturbations are superior when $\frac{-Y}{Z} \geq 2$ may seem somewhat mysterious. It sheds some light on our answer to rewrite (5) as

$$< \Delta E >= -\eta X + \eta^2 (\frac{Y}{P} + Z).$$

For strict gradient descent, the corresponding equation is

$$< \Delta E >= \Delta E = -\eta X + \eta^2 Z.$$

The difference between strict gradient descent and perturbative gradient descent, on average, is the second order term $\eta^2 \frac{Y}{P}$. This is the term which results from not following the gradient exactly, and it obviously goes down as $P$ goes up and the gradient measurement becomes more accurate. Thus, if $Z$ and $Y$ have different signs, $P$ can be used to make the second order term disappear. There is no way to know whether this situation will occur frequently. Furthermore, it is important to keep in mind that if $Y$ is negative and $Z$ is positive, then raising $P$ may make the magnitude of the second order term smaller, but it makes the term itself *larger*. Thus, in general, there is little reason to believe that multiple perturbations will help with a randomly chosen problem.

An example where multiple perturbations help is when we are at a point where the error surface is convex along the gradient direction, and concave in most other directions. Curvature due to second derivative terms in $Y$ and $Z$ help when the gradient direction is followed, but can hurt when we stray from the gradient. In this case, $Z < 0$ and possibly $Y > 0$, so multiple perturbations might be preferable in order to follow the gradient direction very closely.

## 4 SIMULATIONS OF SINGLE AND MULTIPLE PARALLEL PERTURBATION

### 4.1 CONSTANT LEARNING RATES

The second order terms in (5) can be reduced either by using a small learning rate, or by using more perturbations, as discussed briefly in [Cauwenberghs, 1993]. Thus, if $\eta$ is kept constant, we expect a minimum necessary number of perturbations in order to learn. This in itself might be of importance in a limited precision implementation. If there is a non-trivial lower bound on $\eta$, then it might be necessary to use multiple perturbations in order to learn. This is the effect that was noticed in [Alspector et al., 1993]. At that time we thought that we had found empirically

Table 1: Running times for the first initial weight vector

| P | $\eta$ | Time for $< .1$ | Time for $< .5$ |
|---|--------|-----------------|-----------------|
| 1 | .0005 | 1,121,459 | 32,179 |
| 1 | .001 | 831,684 | 18,534 |
| 1 | .002 | 784,768 | 11,008 |
| 1 | .003 | *494,029* | 9,933 |
| 1 | .004 | 1,695,974 | *9,728* |
| 7 | .00625 | 707,840 | 23,834 |
| 7 | .008 | *583,654* | 16,845 |
| 7 | .0125 | 922,880 | 13,261 |
| 7 | .025 | 1,010,355 | *12,006* |
| 7 | .035 | Not tested | 17,024 |

that multiple perturbations were necessary for learning. The problem was that we failed to decrease the learning rate with the number of perturbations.

## 4.2 EMPIRICAL RELATIVE EFFICIENCY OF SINGLE AND MULTIPLE PERTURBATION ALGORITHMS

Section 3 showed that, in theory, multiple perturbations might be faster than 1 perturbation. We investigated whether or not this is the case for the 7 input hamming error correction problem as described in [Biggs, 1989]. This is basically a nearest neighbor problem. There exist 16 distinct 7 bit binary code words. When presented with an arbitrary 7 bit binary word, the network is to output the code word with the least hamming distance from the input.

After preliminary tests with 50, 25, 7, and 1 perturbation, it seemed that 7 perturbations provided the fastest learning, so we concentrated on running simulations for both the 1 perturbation and the 7 perturbation case. Specifically, we chose two different (randomly generated) initial weight vectors, and five different seeds for the pseudo-random function used to generate the $\delta w_i$. For each of these ten cases, we tested both 1 perturbation and 7 perturbations with various learning rates in order to obtain the fastest possible learning.

The 128 possible input patterns were repeatedly presented in order. We investigated how many pattern presentations were necessary to drive the MSE below .1 and how many presentations were necessary to drive it below .5. Recalling the theory developed in section 3, we know that multiple perturbations can be helpful only far away from a stationary point. Thus, we expected that 7 perturbations might be quicker reaching .5 but would be slower reaching .1.

The results are summarized in tables 1 and 2. Each table summarizes information for a different initial weight vector. All of the data presented are averaged over 5 runs, one with each of the different random seeds. The two columns labeled "Time for $< .5$" and "Time for $< .1$" are adjusted according to the assumption that one weight update at 7 perturbations takes 7 times as long as one weight update at 1 perturbation. In each table, the following four numbers appear in italics: the shortest time to reach .1 with 1 perturbation, the shortest time to reach .1 with 7 perturbations, the shortest time to reach .5 with 1 perturbation, and the shortest time to reach .5 with 7 perturbations.

7 perturbations were a loss in three out of four of the experiments. Surprisingly,

Table 2: Running times for the second initial weight vector

| P | $\eta$ | Time for < .1 | Time for < .5 |
|---|---|---|---|
| 1 | .001 | 928,236 | 22,733 |
| 1 | .002 | 719,078 | 12,877 |
| 1 | .003 | 754,739 | 10,675 |
| 1 | .004 | 1,603,354 | 11,750 |
| 7 | .00625 | 629,530 | 27,059 |
| 7 | .008 | 611,610 | 19,712 |
| 7 | .0125 | 912,333 | 15,949 |
| 7 | .025 | 1,580,442 | 14,515 |
| 7 | .035 | Not tested | 17,741 |

the one time that multiple perturbations helped was in reaching .1 from the second initial weight vector. There are several possible explanations for this. To begin with, these learning times are averages over only five simulations each, which makes their statistical significance somewhat dubious. Unfortunately, it was impractical to perform too many experiments as the data obtained required 180 computer simulations, each of which sometimes took more than a day to complete.

Another possible explanation is that .1 may not be "asymptotic enough." The numbers .5 and .1 were chosen somewhat arbitrarily to represent non-asymptotic and asymptotic results. However, there is no way of predicting from the theory how close the error must be to its minimum before asymptotic results become relevant.

The fact that 1 perturbation outperformed 7 perturbations in three out of four cases is not surprising. As explained in section 3, there is in general no reason to believe that multiple perturbations will help on a randomly chosen problem.

## 5 CONCLUSION

Our results show that, under ideal computational conditions, where the learning rate can be adjusted to proper size, that a single parallel perturbation is, except for unusual problems, superior to multiple parallel perturbations. However, under the precision constraints imposed by analog VLSI implementation, where learning rates may not be adjustable and presenting a pattern takes longer than performing a perturbation, multiple parallel perturbations are likely to be the best choice.

**Acknowledgment**

We thank Gert Cauwenberghs and James Spall for valuable and insightful discussions.

## Footnotes

*Present address: Dept. of EECS; MIT; Cambridge, MA 02139; dalippe@mit.edu

[1]If we are at a stationary point then the first term in (5) is 0.

## References

[Alspector et al., 1991] Alspector, J., Gannett, J. W., Haber, S., Parker, M. B., and Chu, R. (1991). A VLSI-efficient technique for generating multiple uncorrelated noise sources and its application to stochastic neural networks. *IEEE Transactions on Circuits and Systems*, 38:109–123.

[Alspector et al., 1993] Alspector, J., Meir, R., Yuhas, B., Jayakumar, A., and Lippe, D. (1993). A parallel gradient descent method for learning in analog

VLSI neural networks. In Hanson, S. J., Cowan, J. D., and Giles, C. L., editors, *Advances in Neural Information Processing Systems 5*, pages 836–844, San Mateo, California. Morgan Kaufmann Publishers.

[Biggs, 1989] Biggs, N. L. (1989). *Discrete Math.* Oxford University Press.

[Brown et al., 1992] Brown, T. X., Tran, M. D., Duong, T., and Thakoor, A. P. (1992). Cascaded VLSI neural network chips: Hardware learning for pattern recognition and classification. *Simulation*, 58(5):340–347.

[Cauwenberghs, 1993] Cauwenberghs, G. (1993). A fast stochastic error-descent algorithm for supervised learning and optimization. In Hanson, S. J., Cowan, J. D., and Giles, C. L., editors, *Advances in Neural Information Processing Systems 5*, pages 244–251, San Mateo, California. Morgan Kaufmann Publishers.

[Cauwenberghs, 1994] Cauwenberghs, G. (1994). *Analog VLSI Autonomous Systems for Learning and Optimization*. PhD thesis, California Institute of Technology.

[Flower and Jabri, 1993] Flower, B. and Jabri, M. (1993). Summed weight neuron perturbation: An $o(n)$ improvement over weight perturbation. In Hanson, S. J., Cowan, J. D., and Giles, C. L., editors, *Advances in Neural Information Processing Systems 5*, pages 212–219, San Mateo, California. Morgan Kaufmann Publishers.

[Jabri and Flower, 1991] Jabri, M. and Flower, B. (1991). Weight perturbation: An optimal architecture and learning technique for analog VLSI feedforward and recurrent multilayer networks. In *Neural Computation 3*, pages 546–565.

[Kirk et al., 1993] Kirk, D., Kerns, D., Fleischer, K., and Barr, A. (1993). Analog VLSI implementation of gradient descent. In Hanson, S. J., Cowan, J. D., and Giles, C. L., editors, *Advances in Neural Information Processing Systems 5*, pages 789–796, San Mateo, California. Morgan Kaufmann Publishers.

[Kushner and Clark, 1978] Kushner, H. and Clark, D. (1978). *Stochastic Approximation Methods for Constrained and Unconstrained Systems*. Springer-Verlag, New York.

[Lippe, 1994] Lippe, D. A. (1994). Parallel, perturbative gradient descent methods for learning in analog VLSI neural networks. Master's thesis, Massachusetts Institute of Technology.

[Rumelhart et al., 1986] Rumelhart, D. E., Hinton, G. E., and Williams, R. J. (1986). Learning internal representations by error propogation. In Rumelhart, D. E. and McClelland, J. L., editors, *Parallel Distributed Processing: Explorations in the Microstructure of Cognition*, page 318. MIT Press, Cambridge, MA.

[Spall, 1992] Spall, J. C. (1992). Multivariate stochastic approximation using a simultaneous perturbation gradient approximation. *IEEE Transactions on Automatic Control*, 37(3):332–341.